# Optimizing Correlation Algorithms for Hardware-based Transient Classification

**R. Timothy Edwards[1], Gert Cauwenberghs[1], and Fernando J. Pineda[2]**

[1]Electrical and Computer Engineering, Johns Hopkins University, Baltimore, MD 21218
[2]Applied Physics Laboratory, Johns Hopkins University, Laurel, MD 20723
*e-mail*: {tim, gert, fernando}@bach.ece.jhu.edu

## Abstract

The performance of dedicated VLSI neural processing hardware depends critically on the design of the implemented algorithms. We have previously proposed an algorithm for acoustic transient classification [1]. Having implemented and demonstrated this algorithm in a mixed-mode architecture, we now investigate variants on the algorithm, using time and frequency channel differencing, input and output normalization, and schemes to binarize and train the template values, with the goal of achieving optimal classification performance for the chosen hardware.

## 1 Introduction

At the NIPS conference in 1996 [1], we introduced an algorithm for classifying acoustic transient signals using template correlation. While many pattern classification systems use template correlation [2], our system differs in directly addressing the issue of efficient implementation in analog hardware, to overcome the area and power consumption drawbacks of equivalent digital systems. In the intervening two years, we have developed analog circuits and built VLSI hardware implementing both the template correlation and the frontend acoustic processing necessary to map the transient signal into a time-frequency representation corresponding to the template [3, 4]. In the course of hardware development, we have been led to reevaluate the algorithm in the light of the possibilities and the limitations of the chosen hardware.

The general architecture is depicted in Figure 1 (a), and excellent agreement between simulations and experimental output from a prototype is illustrated in Figure 1 (b). Issues of implementation efficiency and circuit technology aside, the this paper specifically addresses further improvements in classification performance achievable by algorithmic modifications, tailored to the constraints and strengths of the implementation medium.

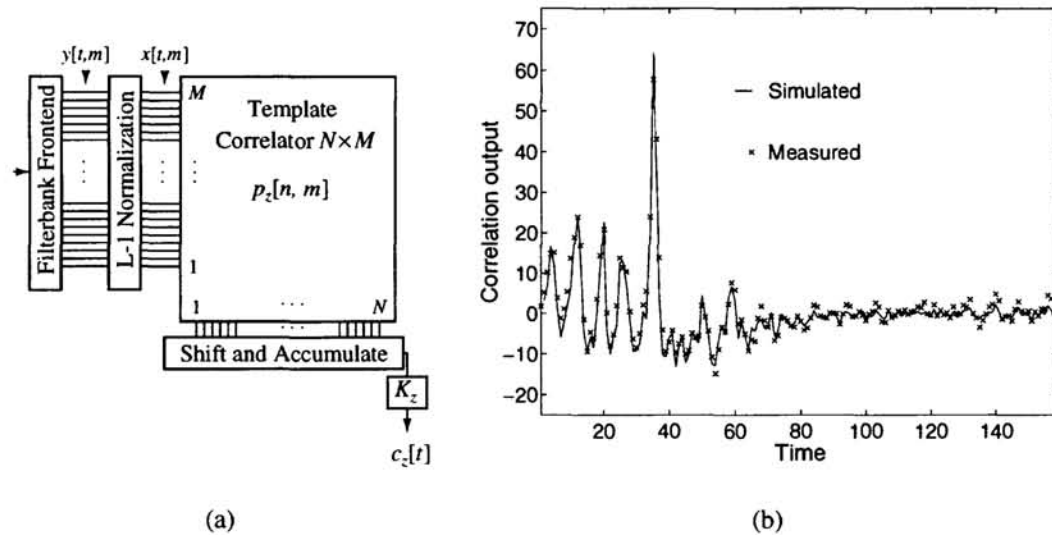

(a)                                                                    (b)

Figure 1: (a) System architecture of the acoustic transient classifier (b) Demonstration of accurate computation in the analog correlator on a transient classification task.

## 2   The transient classification algorithm

The core of our architecture performs the running correlation between an acoustic input and a set of templates for distiguishing between $Z$ distinct classes. A simple template correlation equation for the acoustic transient classification can be written:

$$c_z[t] = K_z \sum_{m=1}^{M} \sum_{n=1}^{N} x[t-n,m]\, p_z[n,m] \tag{1}$$

where $M$ is the number of frequency channels of the input, $N$ is the maximum number of time bins in the window, and $x$ is the array of input signals representing the energy content in each of the $M$ bandpass frequency channels. The inputs $x$ are normalized across channels using an L-1 normalization so that the correlation is less affected by volume changes in the input. The matrix $p_z$ contains the template pattern values for pattern $z$ out of a total of $Z$ classes; $K_z$ is a constant gain coefficient for class $z$, and $t$ is the current time. This formula produces a running correlation $c_z[t]$ of the input array with the template for class $z$. A signal is classified as belonging to class $z$ when the output $c_z$ exceeds the output for all other classes at a point in time $t$ determined by simple segmentation of the input.

To train and evaluate the system, we used a database of 22 recorded samples of 10 different classes of "everyday" transients such as the sounds made by aluminum cans, plastic tubs, handclaps, and the like.

Each example transient recording was processed through a thirty-two channel constant-Q analog cochlear filter with output taps spaced on a logarithmic frequency scale [6]. For the simulations, the frontend system outputs were sampled and saved to disk, then digitally rectified and smoothed with a lowpass filter function with a 2 ms time constant. These thirty-two channel outputs representing short-term average energy in each frequency band were decimated to 500 Hz and normalized with the function

$$x[t,m] = y[t,m] / \sum_{k=1}^{M+1} y[t,k], \tag{2}$$

where $y[t, M+1]$ is a constant-valued input added to the system in order to supress noise in the normalized outputs during periods of silence. The additional output $x[t, M+1]$

becomes maximum during the periods of silence and minimum during presentation of a transient event. This extra output can be used to detect onsets of transients, but is not used in the correlation computation of equation (1).

Template values $p_z$ are learned by automatically aligning all examples of the same class in the training set using a threshold on the normalization output $x[t, M + 1]$, and averaging the values together over $N$ samples, starting a few samples before the point of alignment. Class outputs are normalized relative to one another by multiplying each output by a gain factor $K_z$, computed from the template values using the L-2 norm function

$$K_z = \sqrt{\sum_{m=1}^{M} \sum_{n=1}^{N} p_z[n, m]^2}. \tag{3}$$

We evaluated the accuracy of the system with a cross-validation loop in which we train the system on all of the database except one example of one class, then test on that remaining example, repeating the test for each of the 220 examples in the database. The baseline algorithm gives a classification accuracy of 96.4%.

## 3   Single-bit template values

A major consideration for hardware implementations (both digital and analog) is the memory storage required by the templates, one of which is required for each class. Minimal storage space in terms of bits per template is practical only if the algorithm can be proved to perform acceptably well under decreased levels of quantization of the template values.

At one bit per template location (i.e., $M \times N$ bits per template), the complexity of the hardware is greatly simplified, but it is no longer obvious what method is best to use for learning the template values, or for calculating the per-class gains. The choice of the method is guided by knowledge about the acoustic transients themselves, and simulation to evaluate its effect on the accuracy of a typical classification task.

## 4   Simulations of different zero-mean representations

One bit per template value is a desirable goal, but realizing this goal requires reevaluating the original correlation equation. The input values to be correlated represent band-limited energy spectra, and range from zero to some maximum determined by the L-1 normalization. To determine the value of a template bit, the averaged value over all examples of the class in the training set must be compared to a threshold (which itself must be determined), or else the input itself must be transformed into a form with zero average mean value. In the latter method, the template value is determined by the sign of the transformed input, averaged over all examples of the class in the training set.

The obvious transformations of the input which provide a vector of zero-mean signals to the correlator are the time derivative of each input channel, and the difference between neighboring channels. Certain variations of these are possible, such as a center-surround computation of channel differences, and zero-mean combinations of time and channel differences. While there is evidence that center-surround mechanisms are common to neurobiological signal processing of various sensory modalities in the brain, including processing in the mammalian auditory cortex [5], time derivatives of the input are also plausible in light of the short time base of acoustic transient events. Indeed, there is no reason to assume *a priori* that channel differences are even meaningful on the time scale of transients.

Table 1 shows simulation results, where classification accuracy on the cross-validation test is given for different combinations of continuous-valued and binary inputs and templates,

Table 1: Simulation results with different architectures.

| Method | Both Cont. | Binary Input | Both Binary | Binary $(1,-1)$ Template | Binary $(1,0)$ Template |
|---|---|---|---|---|---|
| One-to-One | 96.40% | — | — | — | — |
| Time Difference | 85.59% | 65.32% | 59.46% | 82.43% | 81.98% |
| Channel Difference | 90.54% | 53.60% | 95.05% | 94.59% | 94.14% |
| Center-Surround | 92.79% | 53.60% | 95.05% | 92.34% | 92.34% |

and different zero-mean transformations of the input. There are several significant points to the results of these classification tasks. The first is to note that in spite of the fact that acoustic transient events are short-term and the time steps between the bins in the template as low as 2 ms, using time differences between samples does not yield reliable classification when either the input or the template or both is reduced to binary form. However, reliability remains high when the correlation is performed using channel differences. The implication is that even the shortest transient events have stable and reliable structure in the frequency domain, a somewhat surprising conclusion given the impulsive nature of most transients.

Another interesting point is that we observe no significant difference between the use of pairwise channel differences and the more complicated center-surround mechanism (twice the channel value minus the value of the two neighboring channels). The slight decrease in accuracy for the center-surround in some instances is most likely due only to the fact that one less channel contributes information to the correlator than in the pairwise channel difference computation. When accuracy is constant, a hardware implementation will always prefer the simpler mechanism.

Very little difference in accuracy is seen between the use of a binary $(1,-1)$ representation and a binary $(1,0)$ representation, in spite of the fact that all zero-valued template positions do not contribute to the correlation output. This lack of difference is a result of the choice of the L-1 normalization across the input vector, which ensures that the part of the correlation due to positive template values is roughly the same magnitude as that due to negative template values, leading to a redundant representation which can be removed without affecting classification results. In analog hardware, particularly current-mode circuits, the $(1,0)$ template representation is much simpler to implement.

Time differencing of the input can be efficiently realized in analog hardware by commuting the time-difference calculation to the end of the correlation computation and implementing it with a simple switch-capacitor circuit. Taking differences between input channel values, on the other hand, is no so easily reduced to a simple hardware form. To find a reasonable solution, we simulated a number of different combinations of channel differencing and binarization. Table 2 shows a few examples. The first row is our standard implementation of channel differences using binary $(1,0)$ templates and continuous-valued input. The drawback of this method in analog hardware is the matching between negative and positive parts of the correlation sum. We found two ways to get around this problem without greatly compromising the system performance: The first, shown in the second row of Table 2 is to add to the correlation sum only if the channel difference is positive and the template value is 1 (one-quadrant multiplication). Another (shown in the last row) is to add the maximum of each pair of channels if the template value is 1, which is preferable in that it uses the input values directly and does not require computing a difference at all. Unfortunately, it also adds a large component to the output which is related only to the total energy of the input and therefore is common to all class outputs, reducing the dynamic range of the system.

Table 2: Simulation results for different methods of computing channel differences

| method | accuracy |
|---|---|
| channel difference | 94.14% |
| one-quadrant multiply | 92.34% |
| maximum channel | 93.69% |

## 5 Optimization of the classifier using per-class gains

The per-class gain values $K_z$ in equation (1) are optimal for the baseline algorithm when using the L-2 normalization. The same normalization applied to the binary templates (when the template value is assumed to be either $+1$ or $-1$) yields the same $K_z$ value for all classes. This unity gain on all class outputs is assumed in all the simulations of the previous section. A careful evaluation of errors from several runs indicated the possibility that different gains on each channel could improve recognition rates, and simple experiments with values tweaked by hand proved this suspicion to be true.

To automate the process of gain optimization, we consider the templates, as determined by averaging together examples of each class in the training set, to be fixed. Then we compute the correlation between each template and the aligned, averaged inputs for each class which were used to generate the templates. The result is a $Z \times Z$ matrix, which we denote $C$, of expected values for the correlation between a typical example of a transient input and the template for its own class (diagonal elements $C_{ii}$) and the templates for all other classes (off-diagonal elements $C_{ij}$, $i \neq j$). Each column of $C$ is like the correlator outputs on which we make a classification decision by choosing the maximum. Therefore we wish to maximize $C_{ii}$ with respect to all other elements in the same column. The only degree of freedom for adjusting these values is to multiply the correlation output of each template $z$ by a constant coefficient $K_z$. This corresponds to multiplying each row of $C$ by $K_z$. This per-class gain mechanism is easily transferred to the analog hardware domain.

In the case of continuous-valued templates, an optimal solution can be directly evaluated and yields the L-2 normalization. However, for all binary forms of the template and/or the input, direct evaluation is impossible and the solution must be found by choosing an error function $\mathcal{E}$ to minimize or maximize. The error function must assign a large error to any off-diagonal element in a column that approaches or exceeds the diagonal element in that column, but must not force the cross-correlations to arbitrarily low negative values. A minimizing function that fits this description is

$$\mathcal{E} = \sum_i \sum_{j \neq i} \exp\left(K_j C_{ji} - K_i C_{ii}\right). \qquad (4)$$

This function unfortunately has no closed-form solution for the coefficients $K_i$, which must be determined numerically using Newton-Raphson or some other iterative method.

Improvements in the recognition rates of the classification task using this optimization of per-class gains is shown in Table 3, where we have considered only the case of inputs and templates encoding channel differences. Although the database is small, the gains of 2 to 4% for the quantized cases are significant. For this particular simulation we used a different type of frontend section to verify that the performance of the correlation algorithm was not linked to a specific frontend architecture. To generate these performance values, we used sixteen channels with the inputs digitally processed through a constant-Q bandpass filter having a Q of 5.0 and with center frequencies spaced on a mel scale from 100 Hz to 4500 Hz. The bandpass filtering was followed by rectification and smoothing with a lowpass filter function with a cutoff frequency scaled logarithmically across channels, from 60 Hz to 600 Hz. The channel output data were decimated to a 500 Hz rate. Half of the

database was used to train the system, and half used to test. Performance is similar to that reported in the previous section in spite of the fact that the number of channels was cut in half, and the number of training examples was also cut in half. Slight gains in performance are most likely due to the cleaner digital filtering of the recorded data.

Table 3: System accuracy with and without per-class normalization.

| binarization | accuracy, optimized | accuracy, non-optimized |
|---|---|---|
| none | 100% | 100% |
| template only | 93% | 91% |
| template & input | 95% | 91% |

## 6   System Robustness

We performed several additional experiment in addition to those covered in the previous sections. One of these was an evaluation of recognition accuracy as a function of the template length $N$ (number of time bins), to determine what is a proper size for the templates. The result is shown in Figure 2 (a). This curve reaches a reliable maximum at about 50 time bins, from which our chosen size for the hardware implementation of 64 bins provides a safe margin of error. However, it is interesting to note that recognition accuracy does not drop to that of random chance until only two time bins are used (64 bits per template), and accuracy is nearly 50% with only 3 time bins (96 bits per template).

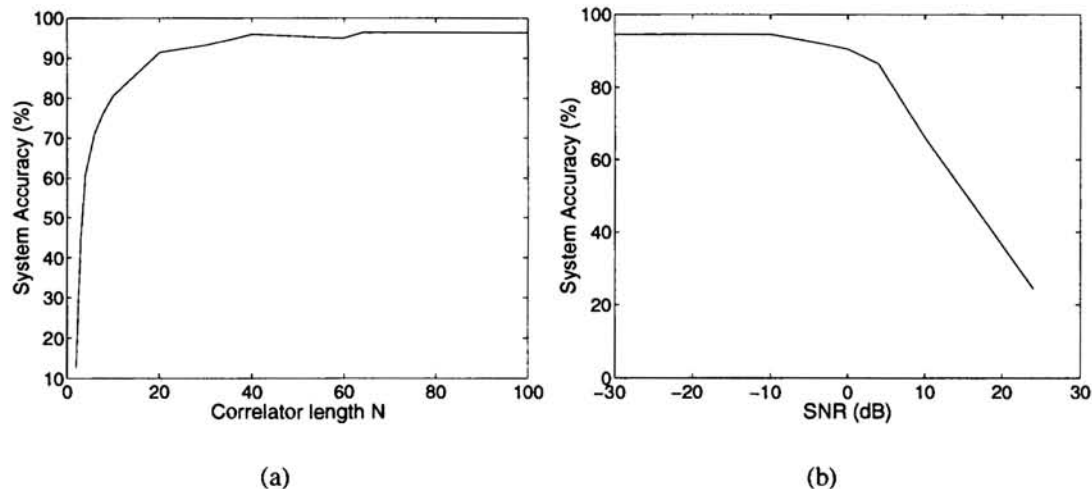

(a)                                          (b)

Figure 2: (a) Effect of decreasing the number of time-bins. (b) Effect of white noise added to the correlator inputs.

We made one evaluation of the robustness of the algorithm in the presence of noise by introducing additional white noise at the correlator inputs. The graph of Figure 2 (right) shows that accuracy remains high until the signal-to-noise ratio is roughly 0 dB.

An interesting question to ask about the L-1 normalization at the frontend is how the added constant normalization channel ($y[t, M + 1]$) affects the classification performance. If this channel is omitted, then the total instantaneous value of all outputs must equal the same value, even during periods of silence, in which low-level noise gets amplified. The nominal value of this channel was chosen to match the levels of noise in the transient recordings. For one of the cases of Table 1 (real input, binary $(1, 0)$ template, channel differencing at

the input), we tried two other tests, one with the normalization constant doubled, and one with it omitted (zero). Doubling the normalization constant had no effect on the error rate, while omitting it caused the accuracy to drop only from 94.1% to 92.3%. The conclusion is that for large templates, random noise has a low probability of producing a spurious positive correlation that would be classified as a transient. The classification algorithm is not largely dependent on input signal normalization.

## 7  Conclusions

Starting from a template correlation architecture for acoustic transient classification targeted for high-density, low-power analog VLSI implementation, we have investigated several variants on the correlation algorithms, accounting for the strengths and constraints of the VLSI implementation medium while maintaining acceptable classification performance.

Reduction of input and templates to binary form does not significantly affect performance, as long as they are transformed to encode the difference in neighboring channels of the original filterbank frontend outputs. This suggests that acoustic transient classification is not only amenable to implementation in simple analog hardware, but also in reasonably simple digital hardware.

In looking for zero-mean representations of the input compatible with a binary template, we found that computing pairwise differences between channels gives a more robust representation than a time-differential form, as was reported previously in [1]. We have found that computing a center-surround function of the inputs yields virtually the same results as taking pairwise channel differences. Where hardware implementation is the goal, the pairwise difference function is preferred due to its greater simplicity.

We have additionally shown that cross-correlations between aligned, averaged inputs and templates can be used with an iterative method to solve for optimal gain coefficients per class output, which yield better classification performance. This is a method which can be applied in general to all template correlation systems.

## References

[1] F. J. Pineda, G. Cauwenberghs, R. T. Edwards, "Bangs, Clicks, Snaps, Thuds, and Whacks: An Architecture for Acoustic Transient Processing," *Neural Information Processing Systems (NIPS)*, Denver, 1996.

[2] K. P. Unnikrishnan, J. J. Hopfield, and D. W. Tank, "Connected-Digit Speaker-Dependent Speech Recognition Using a Neural Network with Time-Delayed Connections," *IEEE Transactions on Signal Processing*, **39**, pp. 698–713, 1991.

[3] R. T. Edwards, G. Cauwenberghs, and F. J. Pineda, "A Mixed-Signal Correlator for Acoustic Transient Classification," *International Symposium on Circuits and Systems (ISCAS)*, Hong Kong, June 1997.

[4] R. T. Edwards and G. Cauwenberghs, "A Second-Order Log-Domain Bandpass Filter for Audio Frequency Applications," *International Symposium on Circuits and Systems (ISCAS)*, Monterey, CA, June 1998.

[5] K. Wang and S. Shamma, "Representation of Acoustic Signals in the Primary Auditory Cortex," *IEEE Trans. Audio and Speech Processing*, 3(5), pp. 382–395, 1995.

[6] F. J. Pineda, K. Ryals, D. Steigerwald, and P. Furth, "Acoustic Transient Processing using the Hopkins Electronic Ear," World Conference on Neural Networks, Washington, D.C., 1995.
